# Using Expectation to Guide Processing:
# A Study of Three Real-World Applications

**Shumeet Baluja**
Justsystem Pittsburgh Research Center &
School of Computer Science, Carnegie Mellon University
baluja@cs.cmu.edu

## Abstract

In many real world tasks, only a small fraction of the available inputs are important at any particular time. This paper presents a method for ascertaining the relevance of inputs by exploiting temporal coherence and predictability. The method proposed in this paper dynamically allocates relevance to inputs by using expectations of their future values. As a model of the task is learned, the model is simultaneously extended to create task-specific predictions of the future values of inputs. Inputs which are either not relevant, and therefore not accounted for in the model, or those which contain noise, will not be predicted accurately. These inputs can be de-emphasized, and, in turn, a new, improved, model of the task created. The techniques presented in this paper have yielded significant improvements for the vision-based autonomous control of a land vehicle, vision-based hand tracking in cluttered scenes, and the detection of faults in the etching of semiconductor wafers.

## 1 Introduction

In many real-world tasks, the extraneous information in the input can be easily confused with the important features, making the specific task much more difficult. One of the methods in which humans function in the presence of many distracting features is to selectively attend to only portions of the input signal. A means by which humans select where to focus their attention is through the use of expectations. Once the important features in the current input are found, an expectation can be formed of what the important features in the next inputs will be, as well as where they will be. The importance of features must be determined in the context of a specific task; different tasks can require the processing of different subsets of the features in the same input.

There are two distinct uses of expectations. Consider Carnegie Mellon's Navlab autonomous navigation system. The road-following module [Pomerleau, 1993] is separate from the obstacle avoidance modules [Thorpe, 1991]. One role of expectation, in which unexpected features are de-emphasized, is appropriate for the road-following module in which the features to be tracked, such as lane-markings, appear in predictable locations. This use of expectation removes distractions from the input scene. The second role of expectation, to emphasize unexpected features, is appropriate for the obstacle avoidance modules. This use of expectation emphasizes unanticipated features of the input scene.

## 2 Architectures for Attention

In many studies of attention, saliency maps (maps which indicate input relevance) have been constructed in a bottom-up manner. For example, in [Koch & Ullman, 1985], a

saliency map, which is not task-specific, is created by emphasizing inputs which are different from their neighbors. An alternate approach, presented in [Clark & Ferrier, 1992], places multiple different, weighted, task-specific feature detectors around the input image. The regions of the image which contain high weighted sums of the detected features are the portion of the scene which are focused upon. Top-down knowledge of which features are used and the weightings of the features is needed to make the procedure task-specific. In contrast, the goal of this study is to learn which task-specific features are relevant without requiring top-down knowledge.

In this study, we use a method based on *Input Reconstruction Reliability Estimation* (IRRE) [Pomerleau, 1993] to determine which portions of the input are important for the task. IRRE uses the hidden units of a neural network (NN) to perform the desired task and to reconstruct the inputs. In its original use, IRRE estimated how confident a network's outputs were by measuring the similarity between the reconstructed and current inputs. Figure 1(Left) provides a schematic of IRRE. Note that the weights between the input and hidden layers are trained to reduce both task and reconstruction error.

Because the weights between the input and hidden layers are trained to reduce both task and reconstruction error, a potential drawback of IRRE is the use of the hidden layer to encode all of the features in the image, rather than only the ones required for solving the particular task [Pomerleau, 1993]. This can be addressed by noting the following: if a strictly layered (connections are only between adjacent layers) feed-forward neural network can solve a given task, the activations of the hidden layer contain, in some form, the important information for this task from the input layer. One method of determining what is contained in the hidden layer is to attempt to reconstruct the original input image, *based solely upon the representation developed in the hidden layer*. Like IRRE, the input image is reconstructed from the activations of the units in the hidden layer. *Unlike IRRE, the hidden units are not trained to reduce reconstruction error, they are only trained to solve the particular task.* The network's allocation of its limited representation capacity at the hidden layer is an indicator of what it deems relevant to the task. *Information which is not relevant to the task will not be encoded in the hidden units.* Since the reconstruction of the inputs is based solely on the hidden units' activations, and the irrelevant portions of the input are not encoded in the hidden units' activations, the inputs which are irrelevant to the task *cannot* be reconstructed. See Figure 1(Right).

By measuring which inputs can be reconstructed accurately, we can ascertain which inputs the hidden units have encoded to solve the task. A synthetic task which demonstrates this idea is described here. Imagine being given a 10x10 input retina such as shown in Figure 2a&b. The task is to categorize many such examples into one of four classes. Because of the random noise in the examples, the simple underlying process, of a cross being present in one of four locations (see Figure 2c), is not easily discernible, although it is the feature on which the classifications are to be based. Given enough examples, the NN will be able to solve this task. However, even after the model of the task is learned, it is difficult to ascertain to which inputs the network is attending. To determine this, we can freeze the weights in the trained network and connect a input-reconstruction layer to the hidden units, as shown in Figure 1(Right). After training these connections, by measuring where the reconstruction matches the actual input, we can determine which inputs the network has encoded in its hidden units, and is therefore attending. See Figure 2d.

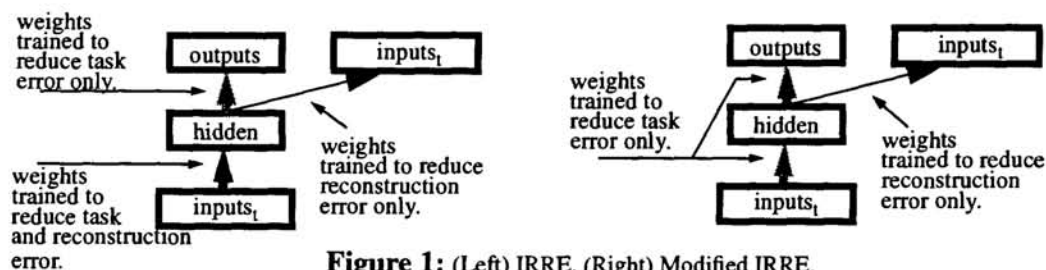

**Figure 1:** (Left) IRRE. (Right) Modified IRRE.

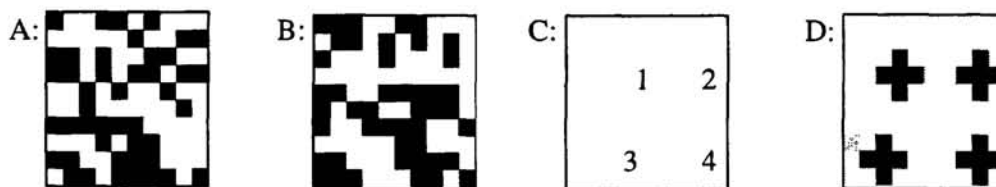

**Figure 2:** (A & B): Samples of training data (cross appears in position 4 & 1 respectively). Note the large amounts of noise. (C): The underlying process puts a cross in one of these four locations. (D): The black crosses are where the reconstruction matched the inputs; these correspond exactly to the underlying process.

IRRE and this modified IRRE are related to auto-encoding networks [Cottrell, 1990] and principal components analysis (PCA). The difference between auto-encoding networks and those employed in this study is that the hidden layers of the networks used here were trained to perform well on the specific task, not to reproduce the inputs accurately.

## 2.1 Creating Expectations

A notion of time is necessary in order to focus attention in future frames. Instead of reconstructing the current input, the network is trained to predict the *next* input; this corresponds to changing the subscript in the reconstruction layer of the network shown in Figure 1(Right) from $t$ to $t+1$. The prediction is trained in a supervised manner, by using the next set of inputs in the time sequence as the target outputs. The next inputs may contain noise or extraneous features. However, since the hidden units only encode information to solve the task, the network will be unable to construct the noise or extraneous features in its prediction.

To this point, a method to create a task-specific expectation of what the next inputs will be has been described. As described in Section 1, there are two fundamentally different ways in which to interpret the difference between the expected next inputs and the actual next inputs. The first interpretation is that the difference between the expected and the actual inputs is a point of interest because it is a region which was not expected. This has applications in anomaly detection; it will be explored in Section 3.2. In the second interpretation, the difference between the expected and actual inputs is considered noise. Processing should be de-emphasized from the regions in which the difference is large. This makes the assumption that there is enough information in the previous inputs to specify what and where the important portions of the next image will be. As shown in the road-following and hand-tracking task, this method can remove spurious features and noise.

## 3 Real-World Applications

Three real-world tasks are discussed in this section. The first, vision-based road following, shows how the task-specific expectations developed in the previous section can be used to eliminate distractions from the input. The second, detection of anomalies in the plasma-etch step of wafer fabrication, shows how expectations can be used to emphasize the unexpected features in the input. The third, visual hand-tracking, demonstrates how to incorporate *a priori* domain knowledge about expectations into the NN.

### 3.1 Application 1: Vision-Based Autonomous Road Following

In the domain of autonomous road following, the goal is to control a robot vehicle by analyzing the image of the road ahead. The direction of travel should be chosen based on the location of important features like lane markings and road edges. On highways and dirt roads, simple techniques, such as feed-forward NNs, have worked well for mapping road images to steering commands [Pomerleau, 1993]. However, on city streets, where there are distractions like old lane-markings, pedestrians, and heavy traffic, these methods fail.

The purpose of using attention in this domain is to eliminate features of the road which the NN may mistake as lane markings. Approximately 1200 images were gathered from a

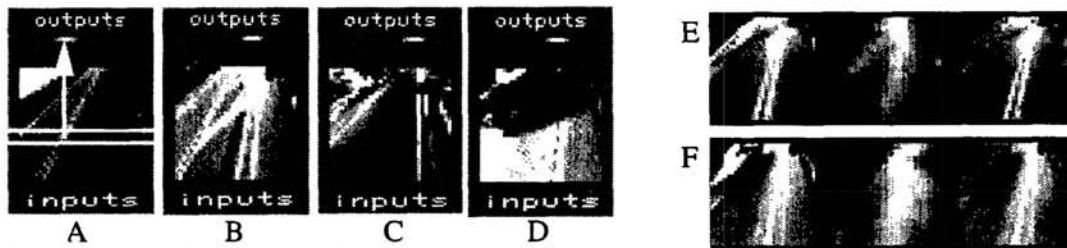

**Figure 3:** (Top): Four samples of training images. Left most shows the position of the lane-marking which was hand-marked. (Right): In each triplet: Left: raw input image$_t$. Middle: the network's prediction of the inputs at time $t$; this prediction was made by a network with input of image$_{t-1}$. Right: a pixel-by-pixel filtered image (see text). This image is used as the input to the NN.

camera mounted on the left side of the CMU-Navlab 5 test vehicle, pointed downwards and slightly ahead of the vehicle. The car was driven through city and residential neighborhoods around Pittsburgh, PA. The images were gathered at 4-5 hz. The images were subsampled to 30x32 pixels. In each of these images, the horizontal position of the lane marking in the 20th row of the input image was manually identified. The task is to produce a Gaussian of activation in the outputs centered on the horizontal position of the lane marking in the 20th row of the image, given the entire input image. Sample images and target outputs are shown in Figure 3. In this task, the ANN can be confused by road edges (Figure 3a), by extraneous lane markings (Figure 3b), and reflections on the car itself (since the camera was positioned on the side of the car), as shown in Figure 3c.

The network architecture shown in Figure 4 was used; this is the same architecture as in Figure 1(right) with the feedback shown. The feedback is used during both training and simulation. In each time-step, a steering direction and a prediction of the next inputs is produced. For each time-step, the magnitude of the difference between the input's expected value (computed in the previous time-step) and its actual value is computed. Each input pixel can be moved towards its *background value*[1] in proportion to this difference-value. The larger the difference value, the more weight is given to the background value. If the difference value is small, the actual inputs are used. This has the effect of de-emphasizing the unexpected inputs.

The results of using this method were very promising. The lane tracker removed distracting features from the images. In Figure 3G, a distracting lane-marking is removed: the lane marker on the right was correctly tracked in images before the distractor lane-marker appeared. In Figure 3F, a passing car is de-emphasized: the network does not have a model to predict the movement of passing cars, since these are not relevant for the lane-marker detection task. In Figure 3E, the side of the road appears brighter than expected; therefore it is de-emphasized. Note that the expectation-images (shown in the middle of each triplet

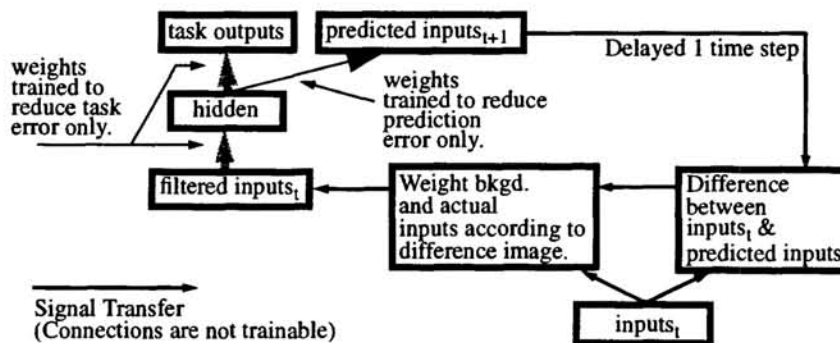

**Figure 4:**
Architecture used to track the lane marking in cluttered scenes.

1. A simple estimate of the background value for each pixel is its average activation across the training set. For the road-following domain, it is possible to use a background activation of 0.0 (when the entire image is scaled to activations of +1.0 to -1.0) since the road often appears as intermediate grays.

in Figure 3) show that the expected lane-marker and road edge locations are not precisely defined. This is due to the training method, which attempts to model the many possible transitions from one time step to the next to account for inter- and intra-driver variability with a limited training set [Baluja, 1996].

In summary, by eliminating the distractions in the input images, the lane-tracker with the attention mechanisms improved performance by 20% over the standard lane-tracker, measured on the difference between the estimated and hand-marked position of the lane-marker in each image. This improvement was seen on multiple runs, with random initial weights in the NN and different random translations chosen for the training images.

### 3.2 Application 2: Fault Detection in the Plasma-Etch Wafer Fabrication

Plasma etch is one of the many steps in the fabrication of semiconductor wafers. In this study, the detection of four faults was attempted. Descriptions of the faults can be found in [Baluja, 1996][Maxion, 1996]. For the experiments conducted here, only a single sensor was used, which measured the intensity of light emitted from the plasma at the 520nm wavelength. Each etch was sampled once a second, providing approximately 140 samples per wafer waveform. The data-collection phase of this experiment began on October 25, 1994, and continued until April 4, 1995. The detection of faults is a difficult problem because the contamination of the etch chamber and the degradation parts keeps the sensor's outputs, even for fault-free wafers, changing over time. Accounting for machine state should help the detection process.

Expectation is used as follows: Given the waveform signature of wafer$_{T-1}$, an expectation of wafer$_T$ can be formed. The input to the prediction-NN is the waveform signature of wafer$_{T-1}$; the output is the prediction of the signature of wafer$_T$. The target output for each example is the signature of the next wafer in sequence (the full 140 parameters). Detection of the four faults is done with a separate network which used as input: the expectation of the wafer's waveform, the actual wafer's waveform, and the point-by-point difference of the two. In this task, the input is not filtered as in the driving domain described previously; the values of the point-by-point difference vector are used as extra inputs.

The performance of many methods and architectures were compared on this task, details can be found in [Baluja, 1996]. The results using the expectation based methods was a 98.7% detection rate, 100% classification rate on the detected faults (determining which of the four types of faults the detected fault was), and a 2.3% false detection rate. For comparison, a simple perceptron had an 80% detection rate, and a 40% false-detection rate. A fully-connected network which did not consider the state of the machine achieved a 100% detection rate, but a 53% false detection rate. A network which considered state by using the last-previous no-fault wafer for comparison with the current wafer (instead of an expectation for the current wafer) achieved an 87.9% detection rate, and a 1.5% false-detection rate. A variety of neural and non-neural methods which examined the differences between the expected and current wafer, as well those which examined the differences between the last no-fault wafer and the current wafer, performed poorly. In summary, methods which did not use expectations were unable to obtain the false-positives and detection rates of the expectation-based methods.

### 3.3 Application 3: Hand-Tracking in Cluttered Scenes

In the tasks described so far, the transition rules were *learned* by the NN. However, if the transition rules had been known *a priori*, processing could have been directed to only the relevant regions by explicitly manipulating the expectations. The ability to incorporate *a priori* rules is important in many vision-based tasks. Often the constraints about the environment in which the tracking is done can be used to limit the portions of the input scene which need to be processed. For example, consider visually tracking a person's hand. Given a fast camera sampling rate, the person's hand in the current frame will be close to

**A.** 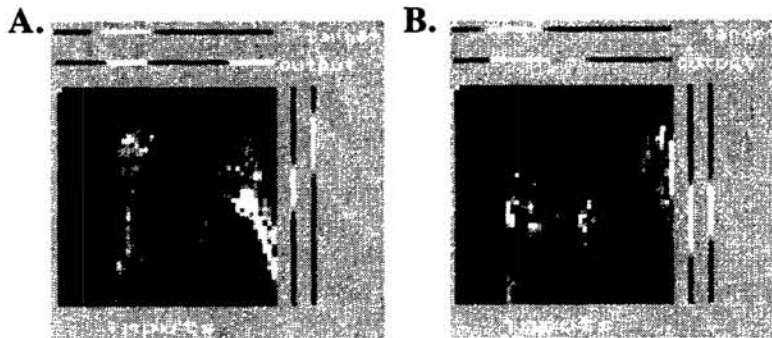 **B.**

**Figure 5:**
Typical input images used for the hand-tracking experiments. The target is to track the subject's right hand. Without expectation, in (A) both hands were found in **X** outputs, and the wrong hand was found in the **Y** outputs. In (B) Subject's right hand and face found in the **X** outputs.

where it appeared in the previous frame. Although a network can learn this constraint by developing expectations of future inputs (as with the NN architecture shown in Figure 4), training the expectations can be avoided by incorporating this rule directly.

In this task, the input layer is a 48*48 image. There are two output layers of 48 units; the desired outputs are two gaussians centered on the (X,Y) position of the hand to be tracked. See Figure 5. Rather than creating a saliency map based upon the difference between the actual and predicted inputs, as was done with autonomous road following, the saliency map was explicitly created with the available domain knowledge. Given the sampling rate of the camera and the size of the hand in the image, the salient region for the next time-step was a circular region centered on the estimated location of the hand in the previous image. The activations of the inputs outside of the salient region were shifted towards the background image. The activations inside the salient region were not modified. After applying the saliency map to the inputs, the filtered inputs were fed into the NN.

This system was tested in very difficult situations; the testing set contained images of a person moving both of his hands and body throughout the sequence (see Figure 5). Therefore, both hands and body are clearly visible in the difference images used as input into the network. All training was done on much simpler training sets in which only a single hand was moving. To gauge the performance of an expectation-based system, it was compared to a system which used the following post-processing heuristics to account for temporal coherence. First, before a gaussian was fit to either of the output layers, the activation of the outputs was inversely scaled with the distance away from the location of the hand in the previous time step. This reduces the probability of detecting a hand in a location very different than the previous detection. This helps when both hands are detected, as shown in Figure 5. The second heuristic was that any predictions which differ from the previous prediction by more than half of the dimension of the output layer were ignored, and the previous prediction used instead. See Table I for the results. In summary, by using the expectation based methods, performance improved from 66% to 90% when tracking the left hand, and 52% to 91% when tracking the right hand.

**Table I: Performance: Number of frames in which each hand was located (283 total images).**

| Method | Target: Find Left Hand | | | | Target: Find Right Hand | | | |
|---|---|---|---|---|---|---|---|---|
| | Which Hand Was Found | | | | Which Hand Was Found | | | |
| | % Correct | L | R | None | % Correct | L | R | None |
| No Heuristics, No Expect. | 52% | 146 | 44 | 93 | 16% | 143 | 47 | 93 |
| Heuristics | 66% | 187 | 22 | 74 | 52% | 68 | 147 | 68 |
| Expectation | 91% | 258 | 3 | 22 | 90% | 3 | 255 | 25 |
| Expectation + Heuristics | 90% | 256 | 3 | 24 | 91% | 2 | 257 | 24 |

[Nowlan & Platt, 1995] presented a convolutional-NN based hand-tracker which used separate NNs for intensity and differences images with a rule-based integration of the multiple network outputs. The integration of this expectation-based system should improve the performance of the difference-image NN.

## 4 Conclusions

A very closely related procedure to the one described in this paper is the use of Kalman Filters to predict the locations of objects of interest in the input retina. For example, Dickmanns uses the prediction of the future state to help guide attention by controlling the direction of a camera to acquire accurate position of landmarks [Dickmanns, 1992]. Strong models of the vehicle motion, the appearance of objects of interest (such as the road, road-signs, and other vehicles), and the motion of these objects are encoded in the system. The largest difference in their system and the one presented here is the amount of *a priori* knowledge that is used. Many approaches which use Kalman Filters require a large amount of problem specific information for creating the models. In the approach presented in this paper, the main object is to automatically learn this information from examples. First, the system must learn what the important features are, since no top-down information is assumed. Second, the system must automatically develop the control strategy from the detected features. Third, the system must also learn a model for the movements of all of the relevant features.

In deciding whether the approaches described in this paper are suitable to a new problem, two criteria must be considered. First, if expectation is to be used to remove distractions from the inputs, then given the current inputs, the activations of the relevant inputs in the next time step must be predictable while the irrelevant inputs are either unrelated to the task or are unpredictable. In many visual object tracking problems, the relevant inputs are often predictable while the distractions are not. In the cases in which the distractions are predictable, if they are unrelated to the main task, these methods can work. When using expectation to emphasize unexpected or potentially anomalous features, the activations of the relevant inputs should be unpredictable while the irrelevant ones are predictable. This is often the case for anomaly/fault detection tasks. Second, when expectations are used as a filter, it is necessary to explicitly define the role of the expected features. In particular, it is necessary to define whether the expected features should be considered relevant or irrelevant, and therefore, whether they should be emphasized or de-emphasized, respectively.

We have demonstrated the value of using task-specific expectations to guide processing in three real-world tasks. In complex, dynamic, environments, such as driving, expectations are used to quickly and accurately discriminate between the relevant and irrelevant features. For the detection of faults in the plasma-etch step of semiconductor fabrication, expectations are used to account for the underlying drift of the process. Finally, for vision-based hand-tracking, we have shown that *a priori* knowledge about expectations can be easily integrated with a hand-detection model to focus attention on small portions of the scene, so that distractions in the periphery can be ignored.

**Acknowledgments**

The author would like to thank Dean Pomerleau, Takeo Kanade, Tom Mitchell and Tomaso Poggio for their help in shaping this work.

**References**

Baluja, S. 1996, *Expectation-Based Selective Attention.* Ph.D. Thesis, School of Computer Science, CMU.

Clark, J. & Ferrier, N (1992), Attentive Visual Servoing, in: *Active Vision.* Blake & Yuille, (MIT Press) 137-154.

Cottrell, G.W., 1990, Extracting Features from Faces using Compression Network, *Connectionist Models,* Morgan Kaufmann 328-337.

Dickmanns, 1992, Expectation-based Dynamic Scene Understanding, in: *Active Vision.* A. Blake & A.Yuille, MIT Press.

Koch, C. & Ullman, S. (1985) "Shifts in Selective Visual Attention: Towards the Underlying Neural Circuitry", in: Human Neurobiology 4 (1985) 219-227.

Maxion, R. (1995) The Semiconductor Wafer Plasma-Etch Data Set.

Nowlan, S. & Platt, J., 1995, "A Convolutional Neural Network Hand Tracker". *NIPS 7.* MIT Press. 901-908.

Pomerleau, D.A., 1993. *Neural Network Perception for Mobile Robot Guidance,* Kluwer Academic.

Thorpe, C., 1991, Outdoor Visual Navigation for Autonomous Robots, in: *Robotics and Autonomous Systems 7.*